# Learning Curves: Asymptotic Values and Rate of Convergence

**Corinna Cortes, L. D. Jackel, Sara A. Solla, Vladimir Vapnik,
and John S. Denker**
AT&T Bell Laboratories
Holmdel, NJ 07733

## Abstract

Training classifiers on large databases is computationally demanding. It is desirable to develop efficient procedures for a reliable prediction of a classifier's suitability for implementing a given task, so that resources can be assigned to the most promising candidates or freed for exploring new classifier candidates. We propose such a practical and principled predictive method. Practical because it avoids the costly procedure of training poor classifiers on the whole training set, and principled because of its theoretical foundation. The effectiveness of the proposed procedure is demonstrated for both single- and multi-layer networks.

## 1 Introduction

Training classifiers on large databases is computationally demanding. It is desirable to develop efficient procedures for a reliable prediction of a classifier's suitability for implementing a given task. Here we describe such a practical and principled predictive method.

The procedure applies to real-life situations with huge databases and limited resources. Classifier selection poses a problem because training requires resources — especially CPU-cycles, and because there is a combinatorical explosion of classifier candidates. Training just a few of the many possible classifiers on the full database might take up all the available resources, and finding a classifier particular suitable for the task requires a search strategy.

**Figure 1**: Test errors as a function of the size of the training set for three different classifiers. A classifier choice based on best test error at training set size $l_0 = 10,000$ will result in an inferior classifier choice if the full database contains more than 15,000 patterns.

The naive solution to the resource dilemma is to reduce the size of the database to $l = l_0$, so that it is feasible to train all classifier candidates. The performance of the classifiers is estimated from an independently chosen test set after training. This makes up one point for each classifier in a plot of the test error as a function of the size $l$ of the training set. The naive search strategy is to keep the best classifier at $l_0$, under the assumption that the relative ordering of the classifiers is unchanged when the test error is extrapolated from the reduced size $l_0$ to the full database size. Such an assumption is questionable and could easily result in an inferior classifier choice as illustrated in Fig. 1.

Our predictive method also utilizes extrapolation from medium sizes to large sizes of the training set, but it is based on several data points obtained at various sizes of the training set in the intermediate size regime where the computational cost of training is low. A change in the representation of the measured data points is used to gain confidence in the extrapolation.

## 2   A Predictive Method

Our predictive method is based on a simple modeling of the learning curves of a classifier. By learning curves we mean the expectation value of the test and training errors as a function of the training set size $l$. The expectation value is taken over all the possible ways of choosing a training set of a given size.

A typical example of learning curves is shown in Fig. 2. The test error is always larger than the training error, but asymptotically they reach a common value, $a$. We model the errors for large sizes of the training set as power-law decays to the

**Figure 2**: Learning curves for a typical classifier. For all finite values of the training set size $l$ the test error is larger than the training error. Asymptotically they converge to the same value $a$.

asymptotic error value, $a$:

$$\mathcal{E}_{\text{test}} = a + \frac{b}{l^\alpha} \qquad \text{and} \qquad \mathcal{E}_{\text{train}} = a - \frac{c}{l^\beta}$$

where $l$ is the size of the training set, and $\alpha$ and $\beta$ are positive exponents. From these two expressions the sum and difference is formed:

$$\mathcal{E}_{\text{test}} + \mathcal{E}_{\text{train}} = 2a + \frac{b}{l^\alpha} - \frac{c}{l^\beta} \qquad (1)$$

$$\mathcal{E}_{\text{test}} - \mathcal{E}_{\text{train}} = \frac{b}{l^\alpha} + \frac{c}{l^\beta} \qquad (2)$$

If we make the assumption $\alpha = \beta$ and $b = c$ the equation (1) and (2) reduce to

$$\mathcal{E}_{\text{test}} + \mathcal{E}_{\text{train}} = 2a$$

$$\mathcal{E}_{\text{test}} - \mathcal{E}_{\text{train}} = \frac{2b}{l^\alpha} \qquad (3)$$

These expressions suggest a *log-log* representation of the sum and difference of the test and training errors as a function of the the training set size $l$, resulting in two straight lines for large sizes of the training set: a constant $\sim \log(2a)$ for the sum, and a straight line with slope $-\alpha$ and intersection $\log(b + c) \sim \log(2b)$ for the difference, as shown in Fig. 3.

The assumption of equal amplitudes $b = c$ of the two convergent terms is a convenient but not crucial simplification of the model. We find experimentally that for classifiers where this approximation does not hold, the difference $\mathcal{E}_{\text{test}} - \mathcal{E}_{\text{train}}$ still forms a straight line in a *log-log*-plot. From this line the sum $s = b + c$ can be extracted as the intersection, as indicated on Fig. 3. The weighted sum

**Figure 3**: Within the validity of the power-law modeling of the test and training errors, the sum and difference between the two errors as a function of training set size give two straight lines in a *log-log*-plot: a constant $\sim \log(2a)$ for the sum, and a straight line with slope $-\alpha$ and intersection $\log(b + c) \sim \log(2b)$ for the difference.

$c \cdot \mathcal{E}_{\text{test}} + b \cdot \mathcal{E}_{\text{train}}$ will give a constant for an appropriate choice of $b$ and $c$, with $b + c = s$.

The validity of the above model was tested on numerous boolean classifiers with linear decision surfaces. In all experiments we found good agreement with the model and we were able to extract reliable estimates of the three parameters needed to model the learning curves: the asymptotic value $a$, and the power $\alpha$, and amplitude $b$ of the power-law decay. An example is shown in Fig. 4, (*left*). The considered task is separation of handwritten digits 0–4 from the digits 5–9. This problem is unrealizable with the given database and classifier.

The simple modeling of the test and training errors of equation (3) is only assumed to hold for large sizes of the training set, but it appears to be valid already at intermediate sizes, as seen in Fig. 4, (*left*). The predictive model suggested here is based on this observation, and it can be illustrated from Fig. 4, (*left*): with test and training errors measured for $l \leq 2560$ it is possible to estimate the two straight lines, extract approximate values for the three parameters which characterize the learning curves, and use the resulting power-laws to extrapolate the learning curves to the full size of the database.

The algorithm for the predictive method is therefore as follows:

1. Measure $\mathcal{E}_{\text{test}}$ and $\mathcal{E}_{\text{train}}$ for intermediate sizes of the training set.

2. Plot $\log(\mathcal{E}_{\text{test}} + \mathcal{E}_{\text{train}})$ and $\log(\mathcal{E}_{\text{test}} - \mathcal{E}_{\text{train}})$ versus $\log l$.

3. Estimate the two straight lines and extract the asymptotic value $a$ the amplitude $b$, and the exponent $\alpha$.

4. Extrapolate the learning curves to the full size of the database.

**Figure 4**:
*Left:* Test of the model for a 256 dimensional boolean classifier trained by minimizing a mean squared error. The sum and difference of the test and training errors are shown as a function of the normalized training set size in a *log-log*-plot (base 10). Each point is the mean with standard deviation for ten different choices of a training set of the given size. The straight line with $\alpha = 1$, corresponding to a $1/l$ decay, is shown as a reference.
*Right:* Prediction of learning curves for a 256 dimensional boolean classifier trained by minimizing a mean squared error. Measured errors for training set size of $l \leq 2560$ are used to fit the two proposed straight lines in a *log-log* plot. The three parameters which characterize the learning curves are extracted and used for extrapolation.

A prediction for a boolean classifier with linear decision surface is illustrated in Fig. 4, (*right*). The prediction is excellent for this type of classifiers because the sum and difference of the test and training errors converge quickly to two straight lines in a *log-log*-plot. Unfortunately, linear decision surfaces are in general not adequate for many real-life applications.

The usefulness of the predictive method proposed here can be judged from its performance on real-life sophisticated multi-layer networks. Fig. 5 demonstrates the validity of the model even for a fully-connected multi-layer network operating in its non-linear regime to implement an unrealizable digit recognition task. Already for intermediate sizes of the training set the sum and difference between the test and training errors are again observed to follow straight lines.

The predictive method was finally tested on sparsely connected multi-layer networks. Fig. 6, (*left*), shows the test and training errors for two networks trained for the recognition of handwritten digits. The network termed "old" is commonly referred to as LeNet [LCBD+90]. The network termed "new" is a modification of LeNet with additional feature maps. The full size of the database is 60,000 patterns,

**Figure 5**: Test of the model for a fully-connected 100-10-10 network. The sum and the difference of the test and training error are shown as a function of the normalized training set size in a *log-log*-plot. Each point is the mean with standard deviation for 20 different choices of a training set of the given size.

a 50-50 % mixture of the NIST[1] training and test sets.

After training on 12,000 patterns it becomes obvious that the new network will outperform the old network when trained on the full database, but we wish to quantify the expected improvement. If our predictive method gives a good quantitative estimate of the new network's test error at 60,000 patterns, we can decide whether three weeks of training should be devoted to the new architecture.

A *log-log*-plot based on the three datapoints from the new network result in values for the three parameters that determine the power-laws used to extrapolate the learning curves of the new network to the full size of the database, as illustrated in Fig. 6, (*right*). The predicted test error at the full size of the database $l = 60,000$ is less than half of the test error for the old architecture, which strongly suggest performing the training on the full database. The result of the full training is also indicated in Fig. 6, (*right*). The good agreement between predicted and measured values illustrates the power and applicability of the predictive method proposed here to real-life applications.

## 3    Theoretical Foundation

The proposed predictive method based on power-law modeling of the learning curves is not just heuristic. A fair amount of theoretical work has been done within the framework of statistical mechanics [SST92] to compute learning curves for simple classifiers implementing unrealizable rules with non-zero asymptotic error value. A key assumption of this theoretical approach is that the number of weights in the network is large.

**Figure 6**:

*Left:* Test (circles) and training (triangles) errors for two networks. The "old" network is what commonly is referred to as LeNet. The network termed "new" is a modification of LeNet with additional feature maps. The full size of the database is 60,000 patterns, and it is a 50-50 % mixture of the NIST training and test set.

*Right:* Test (circles) and training (triangles) errors for the new network. The figure shows the predicted values of the learning curves in the range 20,000 - 60,000 training patterns for the "new" network, and the actually measured values at 60,000 patterns.

The statistical mechanical calculations support a symmetric power-law decay of the expected test and training errors to their common asymptotic value. The power-laws describe the behavior in the large $l$ regime, with an exponent $\alpha$ which falls in the interval $1/2 \leq \alpha \leq 1$. Our numerical observations and modeling of the test and training errors are in agreement with these theoretical predictions.

We have, moreover, observed a correlation between the exponent $\alpha$ and the asymptotic error value $a$ not accounted for by any of the theoretical models considered so far. Fig. 7 shows a plot of the exponent $\alpha$ versus the asymptotic error $a$ evaluated for three different tasks. It appears from this data that the more difficult the target rule, the smaller the exponent, or the slower the learning. A larger generalization error for intermediate training set sizes is in such cases due to the combined effect of a larger asymptotic error and a slower convergence. Numerical results for classifiers of both smaller and larger input dimension support the explanation that this correlation might be due to the finite size of the input dimension of the classifier (here 256).

## 4   Summary

In this paper we propose a practical and principled method for predicting the suitability of classifiers trained on large databases. Such a procedure may eliminate

**Figure 7**: Exponent of extracted power-law decay as a function of asymptotic error for three different tasks. The un-realizability of the tasks, as characterized by the asymptotic error $a$, can be changed by tuning the strength of a weight-decay constraint on the norm of the weights of the classifier.

poor classifiers at an early stage of the training procedure and allow for a more intelligent use of computational resources.

The method is based on a simple modeling of the expected training and test errors, expected to be valid for large sizes of the training set. In this model both error measures are assumed to follow power-law decays to their common asymptotic error value, with the same exponent and amplitude characterizing the power-law convergence.

The validity of the model has been tested on classifiers with linear as well as non-linear decision surfaces. The free parameters of the model are extracted from data points obtained at medium sizes of the training set, and an extrapolation gives good estimates of the test error at large size of the training set.

Our numerical studies of learning curves have revealed a correlation between the exponent of the power-law decay and the asymptotic error rate. This correlation is not accounted for by any existing theoretical models, and is the subject of continuing research.

## Footnotes

[1]National Institute for Standards and Technology, Special Database 3.

# References

[LCBD+90] Y. Le Cun, B. Boser, J. S. Denker, D. Henderson, R. E. Howard, W. Hubbard, and L. D. Jackel. Handwritten digit recognition with a back-propagation network. In *Advances in Neural Information Processing Systems*, volume 2, pages 396–404. Morgan Kaufman, 1990.

[SST92]    H. S. Seung, H. Sompolinsky, and N. Tishby. Statistical mechanics of learning from examples. *Physical Review A*, 45:6056–6091, 1992.
